# Probabilistic principles in unsupervised learning of visual structure: human data and a model

**Shimon Edelman, Benjamin P. Hiles & Hwajin Yang**
Department of Psychology
Cornell University, Ithaca, NY 14853
{*se37,bph7,hy56*}*@cornell.edu*

**Nathan Intrator**
Institute for Brain and Neural Systems
Box 1843, Brown University
Providence, RI 02912
*Nathan_Intrator@brown.edu*

## Abstract

To find out how the representations of structured visual objects depend on the co-occurrence statistics of their constituents, we exposed subjects to a set of composite images with tight control exerted over (1) the conditional probabilities of the constituent fragments, and (2) the value of Barlow's criterion of "suspicious coincidence" (the ratio of joint probability to the product of marginals). We then compared the part verification response times for various probe/target combinations before and after the exposure. For composite probes, the speedup was much larger for targets that contained pairs of fragments perfectly predictive of each other, compared to those that did not. This effect was modulated by the significance of their co-occurrence as estimated by Barlow's criterion. For lone-fragment probes, the speedup in all conditions was generally lower than for composites. These results shed light on the brain's strategies for unsupervised acquisition of structural information in vision.

## 1  Motivation

How does the human visual system decide for which objects it should maintain distinct and persistent internal representations of the kind typically postulated by theories of object recognition? Consider, for example, the image shown in Figure 1, left. This image can be represented as a monolithic hieroglyph, a pair of Chinese characters (which we shall refer to as $A$ and $B$), a set of strokes, or, trivially, as a collection of pixels. Note that the second option is only available to a system previously exposed to various combinations of Chinese characters. Indeed, a principled decision whether to represent this image as $\{AB\}$, $\{A, B\}$ or otherwise can only be made on the basis of prior exposure to related images.

According to Barlow's [1] insight, one useful principle is tallying *suspicious coincidences*: two candidate fragments $A$ and $B$ should be combined into a composite object $AB$ if the probability of their joint appearance $P(A, B)$ is much higher than $P(A)P(B)$, which is the probability expected in the case of their statistical independence. This criterion may be compared to the Minimum Description Length (MDL) principle, which has been previously discussed in the context of object representation [2, 3]. In a simplified form [4], MDL calls for representing $AB$ explicitly as a whole if $P(A, B) \gg P(A)P(B)$, just as the principle of suspicious coincidences does.

While the Barlow/MDL criterion $r \doteq P(A, B) / (P(A)P(B))$ certainly indicates a suspicious coincidence, there are additional probabilistic considerations that may be used in setting the degree of association between $A$ and $B$. One example is the possible perfect predictability of $A$ from $B$ and vice versa, as measured by $minCP \doteq \min\{P(A|B), P(B|A)\}$. If $minCP = 1$, then $A$ and $B$ are perfectly predictive of each other and should really be coded by a single symbol, whereas the MDL criterion may suggest merely that some association between the representation of $A$ and that of $B$ be established. In comparison, if $A$ and $B$ are *not* perfectly predictive of each other ($minCP < 1$), there is a case to be made in favor of coding them separately to allow for a maximally expressive representation, whereas MDL may actually suggest a high degree of association (if $r = P(A, B) / (P(A)P(B)) \gg 1$). In this study we investigated whether the human visual system uses a criterion based on $minCP$ alongside MDL while learning (in an unsupervised manner) to represent composite objects.

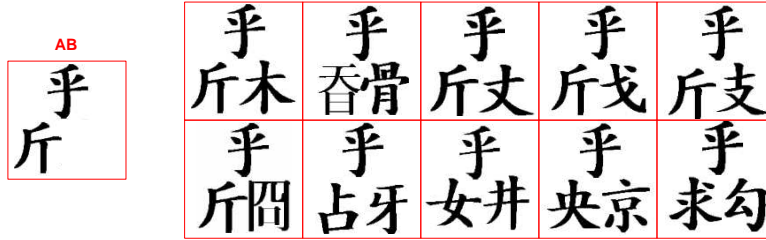

Figure 1: *Left:* how many objects are contained in image $AB$? Without prior knowledge, a reasonable answer, which embodies a holistic bias, should be "one" (Gestalt effects, which would suggest two convex "blobs" [5], are beyond the scope of the present discussion). *Right:* in this set of ten images, $AB$ appears five times as a whole; the other five times a fragment wholly contained in $AB$ appears in isolation. This statistical fact provides grounds for considering $AB$ to be composite, consisting of two fragments (call the upper one $A$ and the lower one $B$), because $P(A|B) = 1$, but $P(B|A) = 0.5 < 1$.

To date, psychophysical explorations of the sensitivity of human subjects to stimulus statistics tended to concentrate on means (and sometimes variances) of the frequency of various stimuli (e.g., [6]. One recent and notable exception is the work of Saffran et al. [7], who showed that infants (and adults) can distinguish between "words" (stable pairs of syllables that recur in a continuous auditory stimulus stream) and non-words (syllables accidentally paired with each other, the first of which comes from one "word" and the second – from the following one). Thus, subjects can sense (and act upon) differences in transition probabilities between successive auditory stimuli. This finding has been recently replicated, with infants as young as 2 months, in the visual sequence domain, using successive presentation of simple geometric shapes with controlled transition probabilities [8]. Also in the visual domain, Fiser and Aslin [9] presented subjects with geometrical shapes in various spatial configurations, and found effects of conditional probabilities of shape co-occurrences, in a task that required the subjects to decide in each trial which of two simultaneously presented shapes was more familiar.

The present study was undertaken to investigate the relevance of the various notions of statistical independence to the unsupervised learning of complex visual stimuli by human subjects. Our experimental approach differs from that of [9] in several respects. First, instead of explicitly judging shape familiarity, our subjects had to verify the presence of a probe shape embedded in a target. This objective task, which produces a pattern of response times, is arguably better suited to the investigation of internal representations involved in object recognition than subjective judgment. Second, the estimation of familiarity requires the subject to access in each trial the representations of all the objects seen in the experi-

ment; in our task, each trial involved just two objects (the probe and the target), potentially sharpening the focus of the experimental approach. Third, our experiments tested the predictions of two distinct notions of stimulus independence: $minCP$, and MDL, or Barlow's ratio.

## 2 The psychophysical experiments

In two experiments, we presented stimuli composed of characters such as those in Figure 1 to nearly 100 subjects unfamiliar with the Chinese script. The conditional probabilities of the appearance of individual characters were controlled. The experiments involved two types of probe conditions: PTYPE=Fragment, or $A \rightarrow ABZ$ (with $V \rightarrow ABZ$ as the reference condition), and PTYPE=Composite, or $AB \rightarrow ABZ$ (with $VW \rightarrow ABZ$ as reference). In this notation (see Figure 2, left), $A$ and $B$ are "familiar" fragments with controlled minimum conditional probability $minCP$, and $V, W, Z$ are novel (low-probability) fragments.

Each of the two experiments consisted of a baseline phase, followed by training exposure (unsupervised learning), followed in turn by the test phase (Figure 2, right). In the baseline and test phases, the subjects had to indicate whether or not the probe was contained in the target (a task previously used by Palmer [5]). In the intervening training phase, the subjects merely watched the character triplets presented on the screen; to ensure their attention, the subjects were asked to note the order in which the characters appeared.

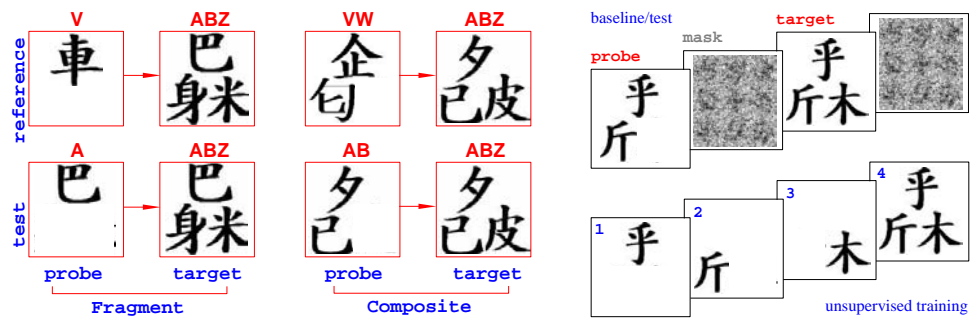

Figure 2: *Left:* illustration of the probe and target composition for the two levels of PTYPE (Fragment and Composite). For convenience, the various categories of characters that appeared in the experiment are annotated here by Latin letters: $A$, $B$ stand for characters with controlled $minCP = \min\{P(A|B), P(B|A)\}$, and $V, W, Z$ stand for characters that appeared only once throughout an experiment. In experiment 1, the training set was constructed with $minCP = 0.5$ for some pairs, and $minCP = 1$ for others; in experiment 2, Barlow's suspicious coincidence ratio $r$ was also controlled. *Right top:* the structure of a part verification trial (same for baseline and test phases). The probe stimulus was followed by the target (each presented for $150\ ms$; a mask was shown before and after the target). The subject had to indicate whether or not the former was contained in the latter (in this example, the correct answer is *yes*). A sequence consisting of 64 trials like this one was presented twice: before training (baseline phase) and after training (test phase). For "positive" trials (i.e., probe contained in target), we looked at the SPEEDUP following training, defined as $RT(baseline) - RT(test)$; negative trials were discarded. *Right bottom:* the structure of a training trial (the training phase, placed between baseline and test, consisted of 80 such trials). The three components of the stimulus appeared one by one for $150\ ms$ to make sure that the subject attended to each, then together for $700\ ms$. The subject was required to note whether the sequence unfolded in a clockwise or counterclockwise order.

The logic behind the psychophysical experiments rested on two premises. First, we knew from earlier work [5] that a probe is detected faster if it is represented monolithically (that is, considered to be a good "object" in the Gestalt sense). Second, we hypothesized that a composite stimulus would be treated as a monolithic object to the extent that its constituent characters are predictable from each other, as measured by a high conditional probability, $minCP$, and/or by a high suspicious coincidence ratio, $r$. The main prediction following from these premises is that the SPEEDUP (the difference in response time between baseline and test phases) for a composite probe should reflect the mutual predictability of the probe's constituents in the training set. Thus, our hypothesis — that *statistics of co-occurrence determine the constituents in terms of which structured objects are represented* — would be supported if the SPEEDUP turns out to be larger for those composite probes whose constituents tend to appear together in the training set. The experiments, therefore, hinged on a comparison of the patterns of response times in the "positive" trials (in which the probe actually *is* embedded in the target; see Figure 2, left) before and after exposure to the training set.

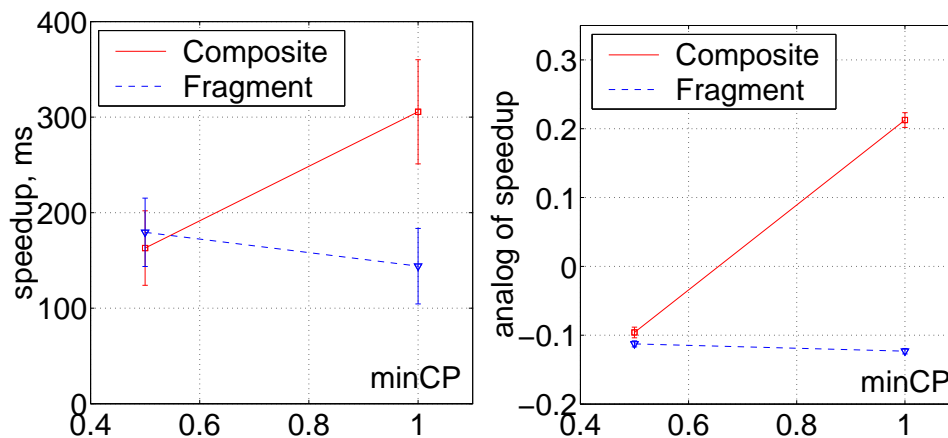

Figure 3: *Left:* unsupervised learning of statistically defined structure by human subjects, experiment 1 ($n = 14$). The dependent variable SPEED-UP is defined as the difference in $RT$ between baseline and test phases (least-squares estimates of means and standard errors, computed by the LSMEANS option of SAS procedure MIXED [10]). The SPEED-UP for composite probes (solid line) with $minCP = 1$ exceeded that in the other conditions by about $150\ ms$. *Right:* the results of a simulation of experiment 1 by a model derived from the one described in [4]. The model was exposed to the same 80 training images as the human subjects. The difference of reconstruction errors for probe and target served as the analog of RT; baseline measurements were conducted on half-trained networks.

## 2.1 Experiment 1

Fourteen subjects, none of them familiar with the Chinese writing system, participated in this experiment in exchange for course credit. Among the stimuli, two characters $A, B$ could be *paired*, in which case we had $P(A|B) = P(B|A) = 1$. Alternatively, $A, B$ could be *unpaired*, with $P(A|B) = 1$, $P(B|A) = 0.5$ (in this experiment, we held the suspicious coincidence ratio $r = P(A, B)/(P(A)P(B))$ constant at $r \approx 8.33$). For the *paired* $A, B$ the minimum conditional probability $minCP = \min\{P(A|B), P(B|A)\} = 1$ and the two characters were perfectly predictable from each other, whereas for the *unpaired* $A, B$ $minCP = 0.5$, and they were not. In the latter case $AB$ probably should not be represented as a whole.

As expected, we found the value of SPEED-UP to be strikingly different for composite probes with $minCP = 1$ (300 $ms$) compared to the other three conditions (about 150 $ms$); see Figure 3, left. A mixed-effects repeated measures analysis of variance (SAS procedure MIXED [10]) for SPEED-UP revealed a marginal effect of PTYPE ($F(1, 13) = 3.80, p < 0.07$) and a significant interaction PTYPE $\times minCP$ interaction ($F(1, 13) = 5.78, p < 0.03$).

This behavior conforms to the predictions of the $minCP$ principle: SPEEDUP was generally higher for composite probes, and disproportionately higher for composite probes with $minCP = 1$. The subjects in experiment 1 proved to be sensitive to the $minCP$ measure of independence in learning to associate object fragments together. Note that the suspicious coincidence ratio was the same in both cases, $r = P(A, B)/(P(A)P(B)) \approx 8.33$. Thus, the visual system is sensitive to $minCP$ over and above the (constant-valued) MDL-related criterion, according to which the propensity to form a unified representation of two fragments, $A$ and $B$, should be determined by $r$ [1, 4].

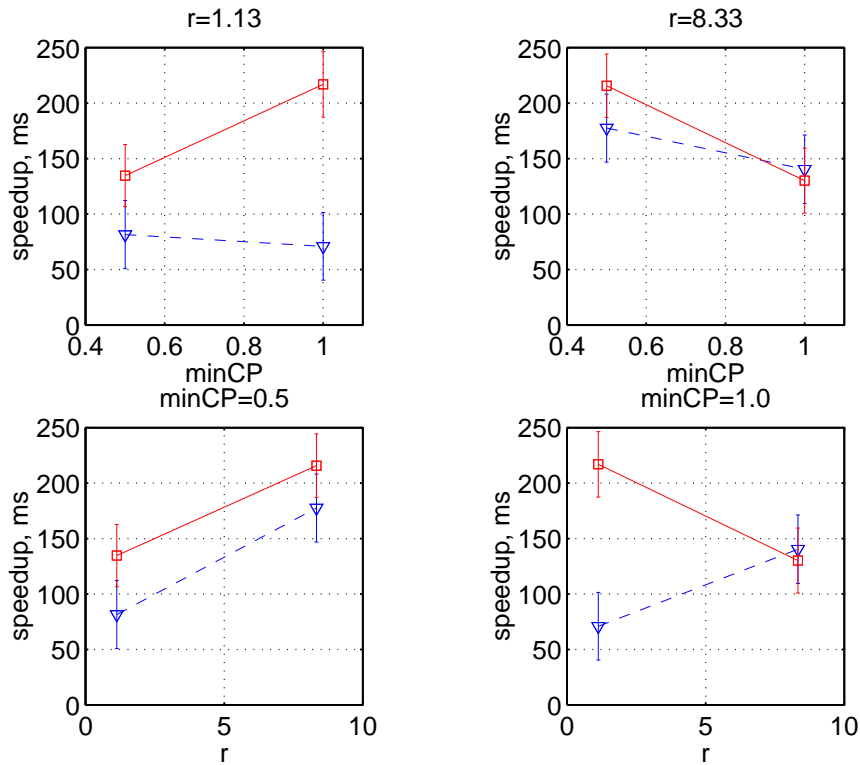

Figure 4: Human subjects, experiment 2 ($n = 81$). The effect of $minCP$ found in experiment 1 was modulated in a complicated fashion by the effect of the suspicious coincidence ratio $r$ (see text for discussion).

## 2.2   Experiment 2

In the second experiment, we studied the effects of varying both $r$ and $minCP$ together. Because these two quantities are related (through the Bayes theorem), they cannot be manipulated independently. To accommodate this constraint, some subjects saw two sets of stimuli, with $minCP = 0.5$, $r = 8.33$ and with $minCP = 1$, $r = 1.13$, in the first ses-

sion and other two sets, with $minCP = 0.5$, $r = 1.13$ and with $minCP = 1$, $r = 8.33$, in the second session; for other subjects, the complementary combinations were used in each session. Eighty one subjects unfamiliar with the Chinese script participated in this experiment for course credit.

The results (Figure 4) showed that SPEEDUP was consistently higher for composite probes. Thus, the association between probe constituents was strengthened by training in each of the four conditions. SPEEDUP was also generally higher for the high suspicious coincidence ratio case, $r = 8.33$, and disproportionately higher for composite probes in the $minCP = 1$, $r = 1.13$ case, indicating a complicated synergy between the two measures of dependence, $minCP$ and $r$. A mixed-effects repeated measures analysis of variance (SAS procedure MIXED [10]) for SPEED-UP revealed significant main effects of PTYPE ($F(1, 78) = 8.92, p < 0.004$) and $r$ ($F(1, 78) = 4.41, p < 0.04$), as well as two significant two-way interactions, $r \times minCP$ ($F(1, 23) = 5.09 p < 0.03$) and $r \times$ PTYPE ($F(1, 75) = 5.08, p < 0.03$). There was also a marginal three-way interaction, $r \times minCP \times$ PTYPE ($F(1, 14) = 3.46, p < 0.08$).

The findings of these two psychophysical experiments can be summarized as follows: (1) an individual complex visual shape (a Chinese character) is detected faster than a composite stimulus (a pair of such characters) when embedded in a 3-character scene, but this advantage is narrowed with practice; (2) a composite attains an "objecthood" status to the extent that its constituents are predictable from each other, as measured either by the conditional probability, $minCP$, or by the suspicious coincidence ratio, $r$; (3) for composites, the strongest boost towards objecthood (measured by response speedup following unsupervised learning) is obtained when $minCP$ is high and $r$ is low, or vice versa. The nature of this latter interaction is unclear, and needs further study.

## 3 An unsupervised learning model and a simulated experiment

The ability of our subjects to construct representations that reflect the probability of co-occurrence of complex shapes has been replicated by a pilot version of an unsupervised learning model, derived from the work of [4]. The model (Figure 5) is based on the following observation: an auto-association network fed with a sequence of composite images in which some fragment/location combinations are more likely than others develops a non-uniform spatial distribution of reconstruction errors. Specifically, smaller errors appear in those locations where the image fragments recur. This information can be used to form a spatial receptive field for the learning module, while the reconstruction error can signal its relevance to the current input [11, 12].

In the simplified pilot model, the spatial receptive field (labeled in Figure 5, left, as "relevance mask") consists of four weights, one per quadrant: $w_i$, $i \in \{1, 2, 3, 4\}$. During the unsupervised training, the weights are updated by setting $w_i^{t+1} = w_i^t / (1 + e^{-(a\Delta \mathbf{x} + b)})$, where $\Delta \mathbf{x}$ is the reconstruction error in trial $t$, and $a$ and $b$ are learning constants. In a simulation of experiment 1, a separate module with its own four-weight "receptive field" was trained for each of the composite stimuli shown to the human subjects.[1] The Euclidean distance between probe and target representations at the output of the model served as the analog of response time, allowing us to compare the model's performance with that of the humans. We found the same differential effects of $minCP$ for `Fragment` and `Composite` probes in the real and simulated experiments; compare Figure 3, left (humans) with Figure 3, right (model).

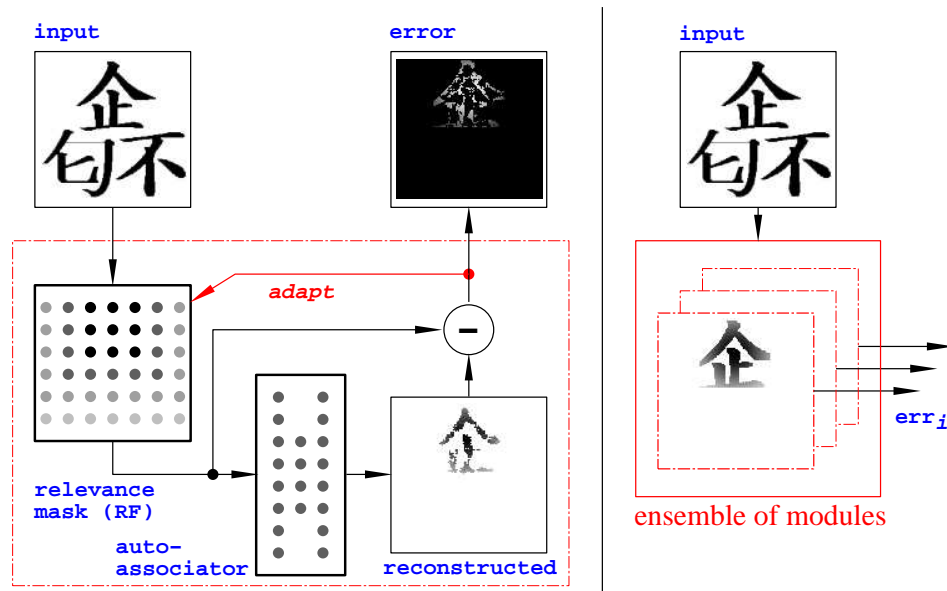

Figure 5: *Left:* the functional architecture of a fragment module. The module consists of two adaptive components: a reconstruction network, and a relevance mask, which assigns different weights to different input pixels. The mask modulates the input multiplicatively, determining the module's receptive field. Given a sequence of images, several such modules working in parallel learn to represent different categories of spatially localized patterns (fragments) that recur in those images. The reconstruction error serves as an estimate of the module's ability to deal with the input ([11, 12]; in the error image, shown on the right, white corresponds to high values). *Right:* the Chorus of Fragments (CoF) is a bank of such fragment modules, each tuned to a particular shape category, appearing in a particular location [13, 4].

## 4 Discussion

Human subjects have been previously shown to be able to acquire, through unsupervised learning, sensitivity to transition probabilities between syllables of nonsense words [7] and between digits [14], and to co-occurrence statistics of simple geometrical figures [9]. Our results demonstrate that subjects can also learn (presumably without awareness; cf. [14]) to treat combinations of complex visual patterns differentially, depending on the conditional probabilities of the various combinations, accumulated during a short unsupervised training session.

In our first experiment, the criterion of suspicious coincidence between the occurrences of $A$ and $B$ was met in both $P(A|B) = 0.5$ and $P(A|B) = 1$ conditions: in each case, we had $r = P(A, B)/(P(A)P(B)) = 8.33$. Yet, the subjects' behavior indicated a significant holistic bias: the representation they form tends to be monolithic ($AB$), unless imperfect mutual predictability of the potential fragments ($A$ and $B$) provides support for representing them separately. We note that a similar holistic bias, operating in a setting where a single encounter with a stimulus can make a difference, is found in language acquisition: an infant faced with an unfamiliar word will assume it refers to the *entire shape* of the most salient object [15]. In our second experiment, both the conditional probabilities as such, and the suspicious coincidence ratio $r$ were found to have the predicted effects, yet these two factors interacted in a complicated manner, which requires a further investigation.

Our current research focuses on (1) the elucidation of the manner in which subjects process statistically structured data, (2) the development of the model of structure learning outlined in the preceding section, and (3) an exploration of the implications of this body of work for wider issues in vision, such as the computational phenomenology of scene perception [16].

## Footnotes

[1]The full-fledged model, currently under development, will have a more flexible receptive field structure, and will incorporate competitive learning among the modules.

## References

[1] H. B. Barlow. Unsupervised learning. *Neural Computation*, 1:295–311, 1989.

[2] R. S. Zemel and G. E. Hinton. Developing population codes by minimizing description length. *Neural Computation*, 7:549–564, 1995.

[3] E. Bienenstock, S. Geman, and D. Potter. Compositionality, MDL priors, and object recognition. In M. C. Mozer, M. I. Jordan, and T. Petsche, editors, *Neural Information Processing Systems*, volume 9. MIT Press, 1997.

[4] S. Edelman and N. Intrator. A productive, systematic framework for the representation of visual structure. In T. K. Leen, T. G. Dietterich, and V. Tresp, editors, *Advances in Neural Information Processing Systems 13*, pages 10–16. MIT Press, 2001.

[5] S. E. Palmer. Hierarchical structure in perceptual representation. *Cognitive Psychology*, 9:441–474, 1977.

[6] M. J. Flannagan, L. S. Fried, and K. J. Holyoak. Distributional expectations and the induction of category structure. *Journal of Experimental Psychology: Learning, Memory and Cognition*, 12:241–256, 1986.

[7] J. R. Saffran, R. N. Aslin, and E. L. Newport. Statistical learning by 8-month-old infants. *Science*, 274:1926–1928, 1996.

[8] N. Z. Kirkham, J. A. Slemmer, and S. P. Johnson. Visual statistical learning in infancy: Evidence for a domain general learning mechanism. *Cognition*, -:–, 2002. in press.

[9] J. Fiser and R. N. Aslin. Unsupervised statistical learning of higher-order spatial structures from visual scenes. *Psychological Science*, 6:499–504, 2001.

[10] SAS. *User's Guide, Version 8*. SAS Institute Inc., Cary, NC, 1999.

[11] D. Pomerleau. Input reconstruction reliability estimation. In C. L. Giles, S. J. Hanson, and J. D. Cowan, editors, *Advances in Neural Information Processing Systems*, volume 5, pages 279–286. Morgan Kaufmann Publishers, 1993.

[12] I. Stainvas and N. Intrator. Blurred face recognition via a hybrid network architecture. In *Proc. ICPR*, volume 2, pages 809–812, 2000.

[13] S. Edelman and N. Intrator. (Coarse Coding of Shape Fragments) + (Retinotopy) $\approx$ Representation of Structure. *Spatial Vision*, 13:255–264, 2000.

[14] G. S. Berns, J. D. Cohen, and M. A. Mintun. Brain regions responsive to novelty in the absence of awareness. *Science*, 276:1272–1276, 1997.

[15] B. Landau, L. B. Smith, and S. Jones. The importance of shape in early lexical learning. *Cognitive Development*, 3:299–321, 1988.

[16] S. Edelman. Constraints on the nature of the neural representation of the visual world. *Trends in Cognitive Sciences*, 6:–, 2002. in press.
